# Online Independent Component Analysis With Local Learning Rate Adaptation

**Nicol N. Schraudolph**
nic@idsia.ch

**Xavier Giannakopoulos**
xavier@idsia.ch

IDSIA, Corso Elvezia 36
6900 Lugano, Switzerland
http://www.idsia.ch/

## Abstract

Stochastic meta-descent (SMD) is a new technique for online adaptation of local learning rates in arbitrary twice-differentiable systems. Like matrix momentum it uses full second-order information while retaining $O(n)$ computational complexity by exploiting the efficient computation of Hessian-vector products. Here we apply SMD to independent component analysis, and employ the resulting algorithm for the blind separation of time-varying mixtures. By matching individual learning rates to the rate of change in each source signal's mixture coefficients, our technique is capable of simultaneously tracking sources that move at very different, *a priori* unknown speeds.

## 1 Introduction

Independent component analysis (ICA) methods are typically run in batch mode in order to keep the stochasticity of the empirical gradient low. Often this is combined with a global learning rate annealing scheme that negotiates the tradeoff between fast convergence and good asymptotic performance. For time-varying mixtures, this must be replaced by a learning rate *adaptation* scheme. Adaptation of a single, global learning rate, however, facilitates the tracking only of sources whose mixing coefficients change at comparable rates [1], *resp.* switch all at the same time [2]. In cases where some sources move much faster than others, or switch at different times, individual weights in the unmixing matrix must adapt at different rates in order to achieve good performance.

We apply stochastic meta-descent (SMD), a new online adaptation method for local learning rates [3, 4], to an extended Bell-Sejnowski ICA algorithm [5] with natural gradient [6] and kurtosis estimation [7] modifications. The resulting algorithm is capable of separating and tracking a time-varying mixture of 10 sources whose unknown mixing coefficients change at different rates.

## 2   The SMD Algorithm

Given a sequence $\vec{x}_0, \vec{x}_1, \ldots$ of data points, we minimize the expected value of a twice-differentiable loss function $f_{\vec{w}}(\vec{x})$ with respect to its parameters $\vec{w}$ by stochastic gradient descent:

$$\vec{w}_{t+1} \;=\; \vec{w}_t \,+\, \vec{p}_t \cdot \vec{\delta}_t\,, \quad \text{where} \quad \vec{\delta}_t \;\equiv\; -\,\frac{\partial f_{\vec{w}_t}(\vec{x}_t)}{\partial \vec{w}} \tag{1}$$

and $\cdot$ denotes component-wise multiplication. The local learning rates $\vec{p}$ are best adapted by exponentiated gradient descent [8, 9], so that they can cover a wide dynamic range while staying strictly positive:

$$\ln \vec{p}_t \;=\; \ln \vec{p}_{t-1} \,-\, \mu\,\frac{\partial f_{\vec{w}_t}(\vec{x}_t)}{\partial \ln \vec{p}}$$

$$\vec{p}_t \;=\; \vec{p}_{t-1} \cdot \exp(\mu\,\vec{\delta}_t \cdot \vec{v}_t)\,, \quad \text{where} \quad \vec{v}_t \;\equiv\; \frac{\partial \vec{w}_t}{\partial \ln \vec{p}} \tag{2}$$

and $\mu$ is a global meta-learning rate. This approach rests on the assumption that each element of $\vec{p}$ affects $f_{\vec{w}}(\vec{x})$ only through the corresponding element of $\vec{w}$. With considerable variation, (2) forms the basis of most local rate adaptation methods found in the literature.

In order to avoid an expensive exponentiation [10] for each weight update, we typically use the linearization $e^u \approx 1 + u$, valid for small $|u|$, giving

$$\vec{p}_t \;=\; \vec{p}_{t-1} \cdot \max(\varrho,\, 1 \,+\, \mu\,\vec{\delta}_t \cdot \vec{v}_t)\,, \tag{3}$$

where we constrain the multiplier to be at least (typically) $\varrho = 0.1$ as a safeguard against unreasonably small — or negative — values. For the meta-level gradient descent to be stable, $\mu$ must in any case be chosen such that the multiplier for $\vec{p}$ does not stray far from unity; under these conditions we find the linear approximation (3) quite sufficient.

**Definition of $\vec{v}$.** The *gradient trace* $\vec{v}$ should accurately measure the effect that a change in local learning rate has on the corresponding weight. It is tempting to consider only the *immediate* effect of a change in $\vec{p}_t$ on $\vec{w}_{t+1}$: declaring $\vec{w}_t$ and $\vec{\delta}_t$ in (1) to be independent of $\vec{p}_t$, one then quickly arrives at

$$\vec{v}_{t+1} \;\equiv\; \frac{\partial \vec{w}_{t+1}}{\partial \ln \vec{p}_t} \;=\; \vec{p}_t \cdot \vec{\delta}_t \tag{4}$$

However, this common approach [11, 12, 13, 14, 15] fails to take into account the incremental nature of gradient descent: a change in $\vec{p}$ affects not only the current update of $\vec{w}$, but also future ones. Some authors account for this by setting $\vec{v}$ to an exponential average of past gradients [2, 11, 16]; we found empirically that the method of Almeida *et al.* [15] can indeed be improved by this approach [3]. While such averaging serves to reduce the stochasticity of the product $\vec{\delta}_t \cdot \vec{\delta}_{t-1}$ implied by (3) and (4), the average remains one of immediate, single-step effects.

By contrast, Sutton [17, 18] models the long-term effect of $\vec{p}$ on future weight updates in a linear system by carrying the relevant partials forward through time, as is done in real-time recurrent learning [19]. This results in an iterative update rule for $\vec{v}$, which we have extended to nonlinear systems [3, 4]. We define $\vec{v}$ as an

exponential average of the effect of *all* past changes in $\vec{p}$ on the current weights:

$$\vec{v}_{t+1} \equiv (1-\lambda) \sum_{i=0}^{\infty} \lambda^i \frac{\partial \vec{w}_{t+1}}{\partial \ln \vec{p}_{t-i}} \tag{5}$$

The *forgetting factor* $0 \leq \lambda \leq 1$ is a free parameter of the algorithm. Inserting (1) into (5) gives

$$
\begin{aligned}
\vec{v}_{t+1} &= \sum_{i=0}^{\infty} \frac{\lambda^i \partial \vec{w}_t}{\partial \ln \vec{p}_{t-i}} + \sum_{i=0}^{\infty} \lambda^i \frac{\partial (\vec{p}_t \cdot \vec{\delta}_t)}{\partial \ln \vec{p}_{t-i}} \\
&\approx \lambda \vec{v}_t + \vec{p}_t \cdot \vec{\delta}_t - \vec{p}_t \cdot \left[ \frac{\partial^2 f_{\vec{w}_t}(\vec{x}_t)}{\partial \vec{w}_t\, \partial \vec{w}_t^T} \sum_{i=0}^{\infty} \frac{\lambda^i \partial \vec{w}_t}{\partial \ln \vec{p}_{t-i}} \right] \\
&= \lambda \vec{v}_t + \vec{p}_t \cdot (\vec{\delta}_t - \lambda H_t \vec{v}_t), \tag{6}
\end{aligned}
$$

where $H_t$ denotes the instantaneous Hessian of $f_{\vec{w}}(\vec{x})$ at time $t$. The approximation in (6) assumes that $(\forall i > 0)\ \partial \vec{p}_t / \partial \vec{p}_{t-i} = 0$; this signifies a certain dependence on an appropriate choice of meta-learning rate $\mu$. Note that there is an efficient $O(n)$ algorithm to calculate $H_t \vec{v}_t$ without ever having to compute or store the matrix $H_t$ itself [20]; we shall elaborate on this technique for the case of independent component analysis below.

**Meta-level conditioning.** The gradient descent in $\vec{p}$ at the meta-level (2) may of course suffer from ill-conditioning just like the descent in $\vec{w}$ at the main level (1); the meta-descent in fact *squares* the condition number when $\vec{v}$ is defined as the previous gradient, or an exponential average of past gradients. Special measures to improve conditioning are thus required to make meta-descent work in non-trivial systems.

Many researchers [11, 12, 13, 14] use the sign function to radically normalize the $\vec{p}$-update. Unfortunately such a nonlinearity does not preserve the zero-mean property that characterizes stochastic gradients in equilibrium — in particular, it will translate any skew in the equilibrium distribution into a non-zero mean change in $\vec{p}$. This causes convergence to non-optimal step sizes, and renders such methods unsuitable for online learning. Notably, Almeida *et al.* [15] avoid this pitfall by using a running estimate of the gradient's stochastic variance as their meta-normalizer.

In addition to modeling the long-term effect of a change in local learning rate, our iterative gradient trace serves as a highly effective conditioner for the meta-descent: the fixpoint of (6) is given by

$$\vec{v}_t = [\lambda H_t + (1-\lambda) \operatorname{diag}(1/\vec{p}_t)]^{-1} \vec{\delta}_t \tag{7}$$

— a modified Newton step, which for typical values of $\lambda$ (*i.e.*, close to 1) scales with the inverse of the gradient. Consequently, we can expect the product $\vec{\delta}_t \cdot \vec{v}_t$ in (2) to be a very well-conditioned quantity. Experiments with feedforward multi-layer perceptrons [3, 4] have confirmed that SMD does not require explicit meta-level normalization, and converges faster than alternative methods.

## 3  Application to ICA

We now apply the SMD technique to independent component analysis, using the Bell-Sejnowski algorithm [5] as our base method. The goal is to find an *unmixing*

matrix $W_t$ which — up to scaling and permutation — provides a good linear estimate $\vec{y}_t \equiv W_t \vec{x}_t$ of the independent sources $\vec{s}_t$ present in a given mixture signal $\vec{x}_t$. The mixture is generated linearly according to $\vec{x}_t = A_t \vec{s}_t$, where $A_t$ is an unknown (and unobservable) full rank matrix.

We include the well-known natural gradient [6] and kurtosis estimation [7] modifications to the basic algorithm, as well as a matrix $P_t$ of local learning rates. The resulting online update for the weight matrix $W_t$ is

$$W_{t+1} = W_t - P_t \cdot D_t, \tag{8}$$

where the gradient $D_t$ is given by

$$D_t \equiv \frac{\partial f_{W_t}(\vec{x}_t)}{\partial W_t} = \left( [\vec{y}_t \pm \tanh(\vec{y}_t)] \, \vec{y}_t^T - I \right) W_t, \tag{9}$$

with the sign for each component of the $\tanh(\vec{y}_t)$ term depending on its current kurtosis estimate.

Following Pearlmutter [20], we now define the differentiation operator

$$\mathcal{R}_{V_t}(g(W_t)) \equiv \left. \frac{\partial g(W_t + r V_t)}{\partial r} \right|_{r=0} \tag{10}$$

which describes the effect on $g$ of a perturbation of the weights in the direction of $V_t$. We can use $\mathcal{R}_{V_t}$ to efficiently calculate the Hessian-vector product

$$H_t \star V_t \equiv \text{vec}^{-1}[H_t \, \text{vec}(V_t)] = \mathcal{R}_{V_t}(D_t) \tag{11}$$

where "vec" is the operator that concatenates all columns of a matrix into a single column vector. Since $\mathcal{R}_{V_t}$ is a linear operator, we have

$$\mathcal{R}_{V_t}(W_t) = V_t, \tag{12}$$

$$\mathcal{R}_{V_t}(\vec{y}_t) = \mathcal{R}_{V_t}(W_t \vec{x}_t) = V_t \vec{x}_t, \tag{13}$$

$$\mathcal{R}_{V_t}(\tanh(\vec{y}_t)) = \text{diag}\big(\tanh'(\vec{y}_t)\big) V_t \vec{x}_t, \tag{14}$$

and so forth (cf. [20]). Starting from (9), we apply the $\mathcal{R}_{V_t}$ operator to obtain

$$
\begin{aligned}
H_t \star V_t &= \mathcal{R}_{V_t}\big[ \left( [\vec{y}_t \pm \tanh(\vec{y}_t)] \, \vec{y}_t^T - I \right) W_t \big] \\
&= \left( [\vec{y}_t \pm \tanh(\vec{y}_t)] \, \vec{y}_t^T - I \right) V_t + \mathcal{R}_{V_t}\big( [\vec{y}_t \pm \tanh(\vec{y}_t)] \, \vec{y}_t^T - I \big) W_t \\
&= \left( [\vec{y}_t \pm \tanh(\vec{y}_t)] \, \vec{y}_t^T - I \right) V_t + \\
&\quad \big[ (I \pm \text{diag}[\tanh'(\vec{y}_t)]) V_t \vec{x}_t \, \vec{y}_t^T + [\vec{y}_t \pm \tanh(\vec{y}_t)](V_t \vec{x}_t)^T \big] W_t
\end{aligned} \tag{15}
$$

In conjunction with the matrix versions of our learning rate update (3)

$$P_t = P_{t-1} \max(\varrho, \, 1 - \mu D_t \cdot V_t) \tag{16}$$

and gradient trace (6)

$$V_{t+1} = \lambda V_t - P_t \cdot (D_t + \lambda H_t \star V_t) \tag{17}$$

this constitutes our SMD-ICA algorithm.

## 4  Experiment

The algorithm was tested on an artificial problem where 10 sources follow elliptic trajectories according to

$$\vec{x}_t = (A_{base} + A_1 \sin(\vec{\omega} t) + A_2 \cos(\vec{\omega} t)) \, \vec{s}_t \qquad (18)$$

where $A_{base}$ is a normally distributed mixing matrix, as well as $A_1$ and $A_2$, whose columns represent the axes of the ellipses on which the sources travel. The velocities $\vec{\omega}$ are normally distributed around a mean of one revolution for every 6 000 data samples. All sources are supergaussian.

The ICA-SMD algorithm was implemented with only online access to the data, including on-line whitening [21]. Whenever the condition number of the estimated whitening matrix exceeded a large threshold (set to 350 here), updates (16) and (17) were disabled to prevent the algorithm from diverging. Other parameters settings were $\mu = 0.1$, $\lambda = 0.999$, and $\rho = 0.2$.

Results that were not separating the 10 sources without ambiguity were discarded. Figure 1 shows the performance index from [6] (the lower the better, zero being the ideal case) along with the condition number of the mixing matrix, showing that the algorithm is robust to a temporary confusion in the separation. The ordinate represents 3 000 data samples, divided into mini-batches of 10 each for efficiency.

Figure 2 shows the match between an actual mixing column and its estimate, in the subspace spanned by the elliptic trajectory. The singularity occurring halfway through is not damaging performance. Globally the algorithm remains stable as long as degenerate inputs are handled correctly.

## 5  Conclusions

Once SMD-ICA has found a separating solution, we find it possible to simultaneously track ten sources that move independently at very different, *a priori* unknown

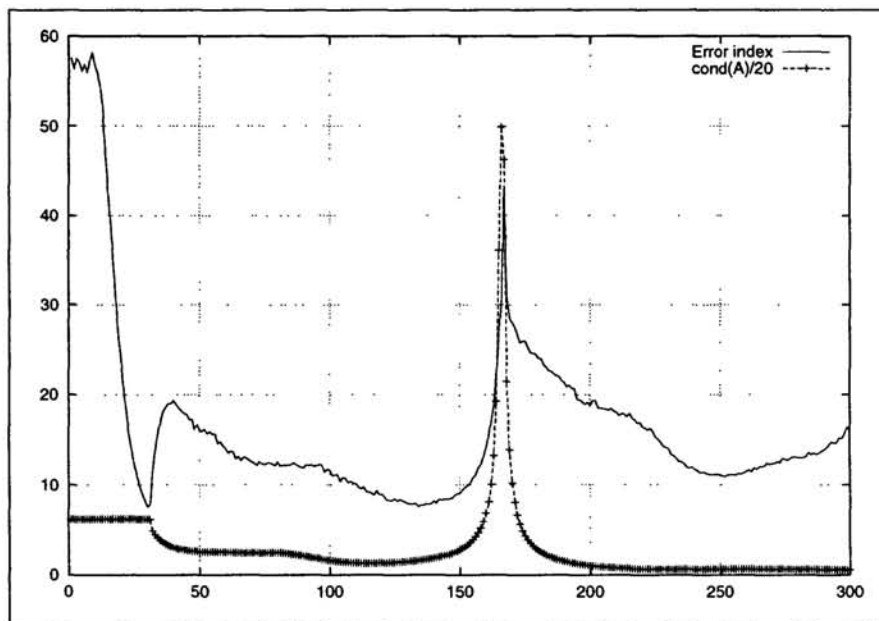

Figure 1: Global view of the quality of separation

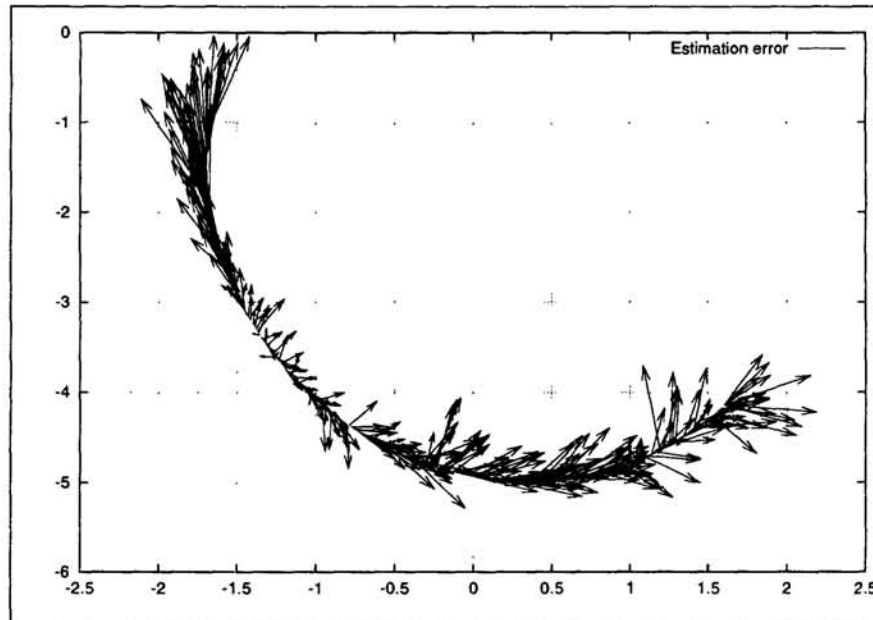

Figure 2: Projection of a column from the mixing matrix. Arrows link the exact point with its estimate; the trajectory proceeds from lower right to upper left.

speeds. To continue tracking over extended periods it is necessary to handle momentary singularities, through online estimation of the number of sources or some other heuristic solution. SMD's adaptation of local learning rates can then facilitate continuous, online use of ICA in rapidly changing environments.

**Acknowledgments**

This work was supported by the Swiss National Science Foundation under grants number 2000–052678.97/1 and 2100–054093.98.

# References

[1] J. Karhunen and P. Pajunen, "Blind source separation and tracking using nonlinear PCA criterion: A least-squares approach", in *Proc. IEEE Int. Conf. on Neural Networks*, Houston, Texas, 1997, pp. 2147–2152.

[2] N. Murata, K.-R. Müller, A. Ziehe, and S.-i. Amari, "Adaptive on-line learning in changing environments", in *Advances in Neural Information Processing Systems*, M. C. Mozer, M. I. Jordan, and T. Petsche, Eds. 1997, vol. 9, pp. 599–605, The MIT Press, Cambridge, MA.

[3] N. N. Schraudolph, "Local gain adaptation in stochastic gradient descent", in *Proceedings of the 9th International Conference on Artificial Neural Networks*, Edinburgh, Scotland, 1999, pp. 569–574, IEE, London, `ftp://ftp.idsia.ch/pub/nic/smd.ps.gz`.

[4] N. N. Schraudolph, "Online learning with adaptive local step sizes", in *Neural Nets — WIRN Vietri-99: Proceedings of the 11th Italian Workshop on Neural Nets*, M. Marinaro and R. Tagliaferri, Eds., Vietri sul Mare, Salerno, Italy, 1999, Perspectives in Neural Computing, pp. 151–156, Springer Verlag, Berlin.

[5] A. J. Bell and T. J. Sejnowski, "An information-maximization approach to blind separation and blind deconvolution", *Neural Computation*, **7**(6):1129–1159, 1995.

[6] S.-i. Amari, A. Cichocki, and H. H. Yang, "A new learning algorithm for blind signal separation", in *Advances in Neural Information Processing Systems*, D. S. Touretzky, M. C. Mozer, and M. E. Hasselmo, Eds. 1996, vol. 8, pp. 757–763, The MIT Press, Cambridge, MA.

[7] M. Girolami and C. Fyfe, "Generalised independent component analysis through unsupervised learning with emergent bussgang properties", in *Proc. IEEE Int. Conf. on Neural Networks*, Houston, Texas, 1997, pp. 1788–1791.

[8] J. Kivinen and M. K. Warmuth, "Exponentiated gradient versus gradient descent for linear predictors", Tech. Rep. UCSC-CRL-94-16, University of California, Santa Cruz, June 1994.

[9] J. Kivinen and M. K. Warmuth, "Additive versus exponentiated gradient updates for linear prediction", in *Proc. 27th Annual ACM Symposium on Theory of Computing*, New York, NY, May 1995, pp. 209–218, The Association for Computing Machinery.

[10] N. N. Schraudolph, "A fast, compact approximation of the exponential function", *Neural Computation*, **11**(4):853–862, 1999.

[11] R. Jacobs, "Increased rates of convergence through learning rate adaptation", *Neural Networks*, **1**:295–307, 1988.

[12] T. Tollenaere, "SuperSAB: fast adaptive back propagation with good scaling properties", *Neural Networks*, **3**:561–573, 1990.

[13] F. M. Silva and L. B. Almeida, "Speeding up back-propagation", in *Advanced Neural Computers*, R. Eckmiller, Ed., Amsterdam, 1990, pp. 151–158, Elsevier.

[14] M. Riedmiller and H. Braun, "A direct adaptive method for faster backpropagation learning: The RPROP algorithm", in *Proc. International Conference on Neural Networks*, San Francisco, CA, 1993, pp. 586–591, IEEE, New York.

[15] L. B. Almeida, T. Langlois, J. D. Amaral, and A. Plakhov, "Parameter adaptation in stochastic optimization", in *On-Line Learning in Neural Networks*, D. Saad, Ed., Publications of the Newton Institute, chapter 6. Cambridge University Press, 1999, ftp://146.193.2.131/pub/lba/papers/adsteps.ps.gz.

[16] M. E. Harmon and L. C. Baird III, "Multi-player residual advantage learning with general function approximation", Tech. Rep. WL-TR-1065, Wright Laboratory, WL/AACF, 2241 Avionics Circle, Wright-Patterson Air Force Base, OH 45433-7308, 1996, http://www.leemon.com/papers/sim_tech/sim_tech.ps.gz.

[17] R. S. Sutton, "Adapting bias by gradient descent: an incremental version of delta-bar-delta", in *Proc. 10th National Conference on Artificial Intelligence*. 1992, pp. 171–176, The MIT Press, Cambridge, MA, ftp://ftp.cs.umass.edu/pub/anw/pub/sutton/sutton-92a.ps.gz.

[18] R. S. Sutton, "Gain adaptation beats least squares?", in *Proc. 7th Yale Workshop on Adaptive and Learning Systems*, 1992, pp. 161–166, ftp://ftp.cs.umass.edu/pub/anw/pub/sutton/sutton-92b.ps.gz.

[19] R. Williams and D. Zipser, "A learning algorithm for continually running fully recurrent neural networks", *Neural Computation*, **1**:270–280, 1989.

[20] B. A. Pearlmutter, "Fast exact multiplication by the Hessian", *Neural Computation*, **6**(1):147–160, 1994.

[21] J. Karhunen, E. Oja, L. Wang, R. Vigario, and J. Joutsensalo, "A class of neural networks for independent component analysis", *IEEE Trans. on Neural Networks*, **8**(3):486–504, 1997.
